# Gaussian Process Latent Variable Models for Visualisation of High Dimensional Data

**Neil D. Lawrence**
Department of Computer Science,
University of Sheffield,
Regent Court, 211 Portobello Street,
Sheffield, S1 4DP, U.K.
neil@dcs.shef.ac.uk

## Abstract

In this paper we introduce a new underlying probabilistic model for principal component analysis (PCA). Our formulation interprets PCA as a particular Gaussian process prior on a mapping from a latent space to the observed data-space. We show that if the prior's covariance function constrains the mappings to be linear the model is equivalent to PCA, we then extend the model by considering less restrictive covariance functions which allow non-linear mappings. This more general Gaussian process latent variable model (GPLVM) is then evaluated as an approach to the visualisation of high dimensional data for three different data-sets. Additionally our non-linear algorithm can be *further* kernelised leading to 'twin kernel PCA' in which a *mapping between feature spaces* occurs.

## 1 Introduction

Visualisation of high dimensional data can be achieved through projecting a data-set onto a lower dimensional manifold. Linear projections have traditionally been preferred due to the ease with which they can be computed. One approach to visualising a data-set in two dimensions is to project the data along two of its principal components. If we were forced to choose *a priori* which components to project along, we might sensibly choose those associated with the largest eigenvalues. The probabilistic reformulation of principal component analysis (PCA) also informs us that choosing the first two components is also the choice that maximises the likelihood of the data [11].

### 1.1 Integrating Latent Variables, Optimising Parameters

Probabilistic PCA (PPCA) is formulated as a latent variable model: given a set centred of $D$-dimensional data $\{\mathbf{y}_n\}_{n=1}^N$ and denoting the latent variable associated with each data-point $\mathbf{x}_n$ we may write the likelihood for an individual data-point under the PPCA model as

$$p\left(\mathbf{y}_n|\mathbf{W},\beta\right) = \int p\left(\mathbf{y}_n|\mathbf{x}_n,\mathbf{W},\beta\right) p\left(\mathbf{x}_n\right) d\mathbf{x}_n$$

where $p(\mathbf{x}_n)$ is Gaussian distributed with unit covariance, $p(\mathbf{x}_n) = N(\mathbf{x}_n|0, \mathbf{I})$, and $p(\mathbf{y}_n|\mathbf{x}_n, \mathbf{W}, \beta) = N(\mathbf{y}_n|\mathbf{W}\mathbf{x}_n, \beta^{-1}\mathbf{I})$. The solution for $\mathbf{W}$ can then be found[1] by assuming that $\mathbf{y}_n$ is i.i.d. and maximising the likelihood of the data-set,

$$p(\mathbf{Y}|\mathbf{W}, \beta) = \prod_{n=1}^{N} p(\mathbf{y}_n|\mathbf{W}, \beta),$$

where $\mathbf{Y} = [\mathbf{y}_1 \dots \mathbf{y}_N]^{\mathrm{T}}$ is the $N \times D$ design matrix.

Probabilistic principal component analysis and other latent variable models, such as factor analysis (FA) or independent component analysis (ICA), require a marginalisation of the latent variables and optimisation of the parameters. In this paper we consider the dual approach of marginalising $\mathbf{W}$ and optimising each $\mathbf{x}_n$. This probabilistic model also turns out to be equivalent to PCA.

### 1.2 Integrating Parameters, Optimising Latent Variables

By first specifying a prior distribution, $p(\mathbf{W}) = \prod_{i=1}^{D} N(\mathbf{w}_i|0, \alpha^{-1}\mathbf{I})$ where $\mathbf{w}_i$ is the $i$th row of the matrix $\mathbf{W}$, and integrating over $\mathbf{W}$ we obtain a marginalised likelihood for $\mathbf{Y}$,

$$p(\mathbf{Y}|\mathbf{X}, \beta) = \frac{1}{(2\pi)^{\frac{DN}{2}} |\mathbf{K}|^{\frac{D}{2}}} \exp\left(-\frac{1}{2}\mathrm{tr}\left(\mathbf{K}^{-1}\mathbf{Y}\mathbf{Y}^{\mathrm{T}}\right)\right), \tag{1}$$

where $\mathbf{K} = \alpha\mathbf{X}\mathbf{X}^{\mathrm{T}} + \beta^{-1}\mathbf{I}$ and $\mathbf{X} = [\mathbf{x}_1 \dots \mathbf{x}_N]^{\mathrm{T}}$. The corresponding log-likelihood is then

$$L = -\frac{DN}{2}\ln(2\pi) - \frac{D}{2}\ln|\mathbf{K}| - \frac{1}{2}\mathrm{tr}\left(\mathbf{K}^{-1}\mathbf{Y}\mathbf{Y}^{\mathrm{T}}\right). \tag{2}$$

Now that the parameters are marginalised we may focus on optimisation of the likelihood with respect to the $\mathbf{X}$. The gradients of (2) with respect to $\mathbf{X}$ may be found as,

$$\frac{\partial L}{\partial \mathbf{X}} = \alpha\mathbf{K}^{-1}\mathbf{Y}\mathbf{Y}^{\mathrm{T}}\mathbf{K}^{-1}\mathbf{X} - \alpha D\mathbf{K}^{-1}\mathbf{X},$$

which implies that at our solution

$$\frac{1}{D}\mathbf{Y}\mathbf{Y}^{\mathrm{T}}\mathbf{K}^{-1}\mathbf{X} = \mathbf{X},$$

some algebraic manipulation of this formula [11] leads to

$$\mathbf{X} = \mathbf{U}_q\mathbf{L}\mathbf{V}^{\mathrm{T}}$$

where $\mathbf{U}_q$ is an $N \times q$ matrix ($q$ is the dimension of the latent space) whose columns are eigenvectors of $\mathbf{Y}\mathbf{Y}^{\mathrm{T}}$, $\mathbf{L}$ is a $q \times q$ diagonal matrix whose $j$th element is $l_j = \left(\frac{\lambda_j}{\alpha D} - \frac{1}{\beta\alpha}\right)^{-\frac{1}{2}}$, where $\lambda_j$ is the $j$th eigenvalue of $\mathbf{Y}\mathbf{Y}^{\mathrm{T}}$, and $\mathbf{V}$ is an arbitrary $q \times q$ orthogonal matrix[2]. Note that the eigenvalue problem we have developed can easily be shown to be equivalent to that solved in PCA (see *e.g.* [10]), indeed the formulation of PCA in this manner is a key step in the development of kernel PCA [9] where $\mathbf{Y}\mathbf{Y}^{\mathrm{T}}$ is replaced with a kernel. Our probabilistic PCA model shares an underlying structure with [11] but differs in that where they optimise we marginalise and where they marginalise we optimise. The marginalised likelihood we are optimising in (1) is recognised as the product of $D$ independent Gaussian processes where the (linear) covariance function is given by $\alpha\mathbf{X}\mathbf{X}^{\mathrm{T}} + \beta^{-1}\mathbf{I}$. Therefore a natural extension is the non-linearisation of the mapping from latent space to the data space through the introduction of a non-linear covariance function.

## 2 Gaussian Process Latent Variable Models

We saw in the previous section how PCA can be interpreted as a Gaussian process 'mapping[3]' from a latent space to a data space where the locale of the points in latent space is determined by maximising the Gaussian process likelihood with respect to $\mathbf{X}$. We will refer to models of this class as Gaussian process latent variable models (GPLVM). Principal component analysis is a GPLVM where the process prior is based on the $N \times N$ inner product matrix of $\mathbf{X}$, in this section we develop an alternative GPLVM by considering a prior which allows for non-linear processes, specifically we focus on the popular 'RBF kernel' which takes the form

$$k_{n,m} = \alpha \exp\left(-\frac{\gamma}{2}\left(\mathbf{x}_n - \mathbf{x}_m\right)^{\mathrm{T}}\left(\mathbf{x}_n - \mathbf{x}_m\right)\right) + \delta_{nm}\beta^{-1}$$

where $k_{n,m}$ is the element in the $n$th row and $m$th column of $\mathbf{K}$, $\gamma$ is a scale parameter and $\delta_{nm}$ denotes the Kronecker delta. Gradients of (2) with respect to the latent points can be found through combining

$$\frac{\partial L}{\partial \mathbf{K}} = \mathbf{K}^{-1}\mathbf{Y}\mathbf{Y}^{\mathrm{T}}\mathbf{K}^{-1} - D\mathbf{K}^{-1},$$

with $\frac{\partial \mathbf{K}}{\partial x_{n,j}}$ via the chain rule. These gradients may be used in combination with (2) in a non-linear optimiser such as scaled conjugate gradients (SCG) [7] to obtain a latent variable representation of the data. Furthermore gradients with respect to the parameters of the kernel matrix may be computed and used to jointly optimise $\mathbf{X}$, $\alpha$, $\gamma$ and $\beta$. The solution for $\mathbf{X}$ will naturally not be unique; even for the linear case described above the solution is subject to an arbitrary rotation, here we may expect multiple local minima.

### 2.1 Illustration of GPLVM via SCG

To illustrate a simple Gaussian process latent variable model we turn to the 'multi-phase oil flow' data [2]. This is a twelve dimensional data-set containing data of three known classes corresponding to the phase of flow in an oil pipeline: stratified, annular and homogeneous. In this illustration, for computational reasons, the data is sub-sampled to 100 data-points.

Figure 1 shows visualisations of the data using both PCA and our GPLVM algorithm which required 766 iterations of SCG. The $\mathbf{X}$ positions for the GPLVM model were initialised using PCA (see `http://www.dcs.shef.ac.uk/~neil/gplvm/` for the MATLAB code used).

The gradient based optimisation of the RBF based GPLVM's latent space shows results which are clearly superior (in terms of greater separation between the different flow domains) to those achieved by the linear PCA model. Additionally the use of a Gaussian process to perform our 'mapping' means that there is uncertainty in the positions of the points in the *data* space. For our formulation the level of uncertainty is shared across all[4] $D$ dimensions and thus may be visualised in the latent space. In Figure 1 (and subsequently) this is done through varying the intensity of the background pixels.

Unfortunately, a quick analysis of the complexity of the algorithm shows that each gradient step requires an inverse of the kernel matrix, an $O\left(N^3\right)$ operation, rendering the algorithm impractical for many data-sets of interest.

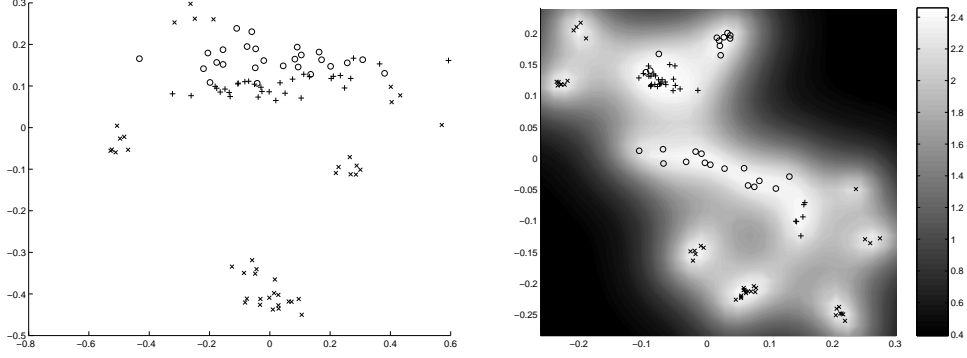

Figure 1: Visualisation of the Oil data with (a) PCA (a linear GPLVM) and (b) A GPLVM which uses an RBF kernel. Crosses, circles and plus signs represent stratified, annular and homogeneous fbws respectively. The greyscales in plot (b) indicate the precision with which the manifold was expressed in data-space for that latent point. The optimised parameters of the kernel were $\gamma = 150$, $\alpha = 0.403$ and $\beta = 316$.

## 2.2 A Practical Algorithm for GPLVMs

There are three main components to our revised, computationally efficient, optimisation process:

**Sparsification.** Kernel methods may be sped up through sparsification, *i.e.* representing the data-set by a subset, $I$, of $d$ points known as the *active set*. The remainder, the *inactive set*, is denoted by $J$. We make use of the informative vector machine [6] which selects points sequentially according to the reduction in the posterior process's entropy that they induce.

**Latent Variable Optimisation.** A point from the inactive set, $j$, can be shown to project into the data space as a Gaussian distribution

$$p\left(\mathbf{y}_j | \mathbf{x}_j, \alpha, \beta, \gamma\right) = N\left(\mathbf{y}_j | \mathbf{f}_j, \sigma_j^2 \mathbf{I}\right) \tag{3}$$

whose mean is $\mathbf{f}_j = \mathbf{Y}^{\mathrm{T}} \mathbf{K}_{I,I}^{-1} \mathbf{k}_{I,j}$ where $\mathbf{K}_{I,I}$ denotes the kernel matrix developed from the active set and $\mathbf{k}_{I,j}$ is a column vector consisting of the elements from the $j$th column of $\mathbf{K}$ that correspond to the active set. The variance is $\sigma_j^2 = k\left(\mathbf{x}_j, \mathbf{x}_j\right) - \mathbf{k}_{I,j}^{\mathrm{T}} \mathbf{K}_{I,I}^{-1} \mathbf{k}_{I,j}$. Note that since $\mathbf{x}_j$ does not appear in the inverse, gradients with respect to $\mathbf{x}_j$ do not depend on other data in $J$. We can therefore independently optimise the likelihood of each $\mathbf{y}_j$ with respect to each $\mathbf{x}_j$. Thus the full set $\mathbf{X}_J$ can be optimised with one pass through the data.

**Kernel Optimisation.** The likelihood of the active set is given by

$$p\left(\mathbf{Y}_I\right) = \frac{1}{(2\pi)^{\frac{D}{2}} |\mathbf{K}_{I,I}|^{\frac{1}{2}}} \exp\left(-\frac{1}{2} \mathbf{Y}_I^{\mathrm{T}} \mathbf{K}_{I,I}^{-1} \mathbf{Y}_I\right), \tag{4}$$

which can be optimised[5] with respect to $\alpha$, $\beta$ and $\gamma$ with gradient evaluations costing $O\left(d^3\right)$.

Algorithm 1 summarises the order in which we implemented these steps. Note that whilst we never optimise points in the active set, we repeatedly reselect the active set so it is

**Algorithm 1** An algorithm for modelling with a GPLVM.

**Require:** A size for the active set, $d$. A number of iterations, $T$.
  Initialise $\mathbf{X}$ through PCA.
  **for** $T$ iterations **do**
    Select a new active set using the IVM algorithm.
    Optimise (4) with respect to the parameters of $\mathbf{K}$ using scaled conjugate gradients.
    Select a new active set.
    **for** Each point not in active set, $j$. **do**
      Optimise (3) with respect to $\mathbf{x}_j$ using scaled conjugate gradients.
    **end for**
  **end for**

unlikely that many points remain in their original location. For all the experiments that follow we used $T = 15$ iterations and an active set of size $d = 100$. The experiments were run on a 'one-shot' basis[6] so we cannot make statements as to the effects that significant modification of these parameters would have. We present results on three data-sets: for the *oil flow data* (Figure 2) from the previous section we now make use of all 1000 available points and we include a comparison with the generative topographic mapping (GTM) [4].

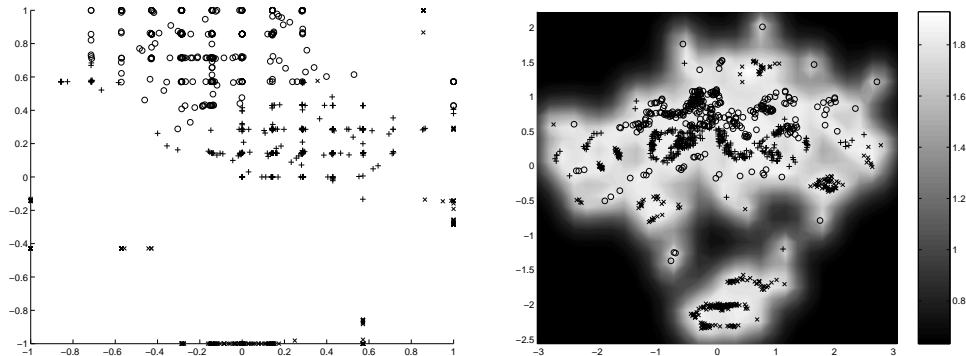

Figure 2: The full oil flow data-set visualised with (a) GTM with 225 latent points laid out on a $15 \times 15$ grid and with 16 RBF nodes and (b) an RBF based GPLVM. The parameters of the latent variable model were found to be $\alpha = 0.225$, $\beta = 128$ and $\gamma = 7.74$. Notice how the GTM artificially 'discretises' the latent space around the locations of the 225 latent points.

We follow [5] in our 2-D visualisation of a sub-set of 3000 of the digits 0-4 (600 of each digit) from a $16 \times 16$ greyscale version of the USPS digit data-set (Figure 3).

Finally we modelled a face data-set [8] consisting of 1965 images from a video sequence digitised at $28 \times 20$. Since the images are originally from a video sequence we might expect the underlying dimensionality of the data to be one — the images are produced in a smooth way over time which can be thought of as a piece of string embedded in a high (560) dimensional pixel space. We therefore present ordered results from a 1-D visualisation in Figure 4 .

All the code used for performing the experiments is available from `http://www.dcs.`

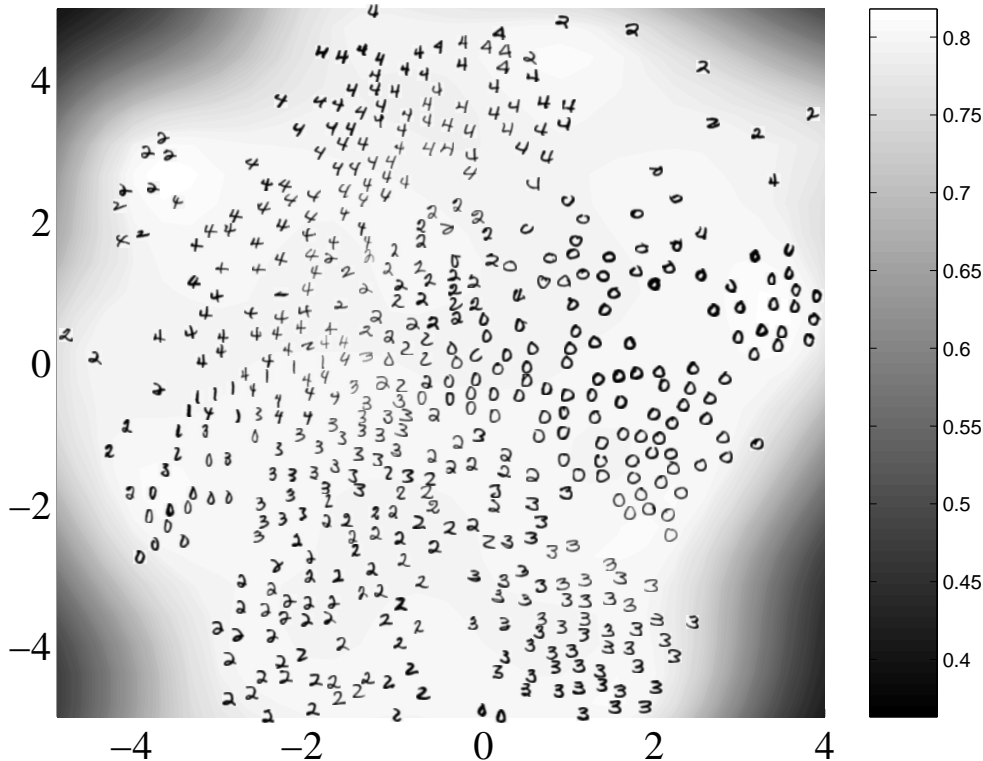

Figure 3: The digit images visualised in the 2-D latent space. We followed [5] in plotting images in a random order but not plotting any image which would overlap an existing image. 538 of the 3000 digits are plotted. Note how little space is taken by the 'ones' (the thin line running from (-4, -1.5) to (-1, 0)) in our visualisation, this may be contrasted with the visualisation of a similar data-set in [5]. We suggest this is because 'ones' are easier to model and therefore do not require a large region in latent space.

`shef.ac.uk/~neil/gplvm/` along with avi video files of the 1-D visualisation and results from two further experiments on the same data (a 1-D GPLVM model of the digits and a 2-D GPLVM model of the faces).

## 3    Discussion

Empirically the RBF based GPLVM model gives useful visualisations of a range of data-sets. Strengths of the method include the ability to *optimise the kernel parameters* and to *generate fantasy data* from any point in latent space. Through the use of a probabilistic process we can obtain *error bars* on the position of the manifolds which can be visualised by imposing a greyscale image upon the latent space.

**When Kernels Collide: Twin Kernel PCA**    The eigenvalue problem which provides the maxima of (2) with respect to $\mathbf{X}$ for the linear kernel is exploited in kernel PCA. One could consider a 'twin kernel' PCA where both $\alpha \mathbf{X}\mathbf{X}^{\mathrm{T}} + \beta^{-1}\mathbf{I}$ and $\mathbf{Y}\mathbf{Y}^{\mathrm{T}}$ are replaced by kernel functions. Twin kernel PCA could no longer be undertaken with an eigenvalue decomposition but Algorithm 1 would still be a suitable mechanism with which to determine the values of $\mathbf{X}$ and the parameters of $\mathbf{X}$'s kernel.

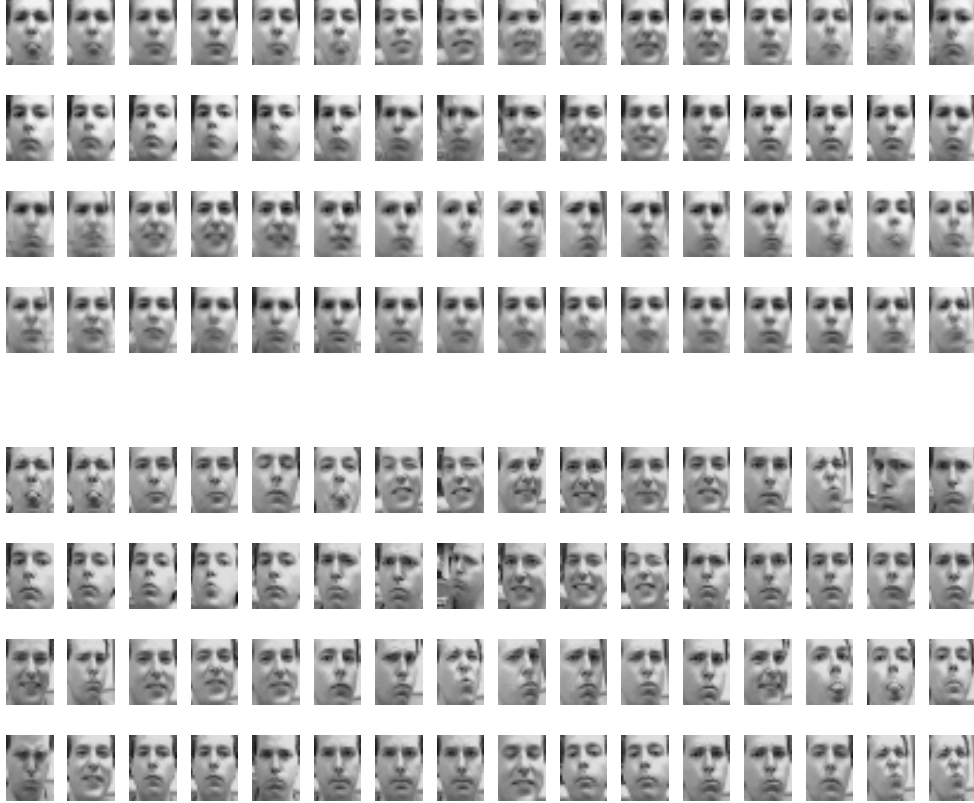

Figure 4: *Top*: Fantasy faces from the 1-D model for the face data. These faces were created by taking 64 uniformly spaced and ordered points from the latent space and visualising the mean of their distribution in data space. The plots above show this sequence unfolding (starting at the top left and moving right). Ideally the transition between the images should be smooth. *Bottom*: Examples from the data-set which are closest to the corresponding fantasy images in *latent* space. Full sequences of 2000 fantasies and the entire dataset are available on the web as avi fi les.

**Stochastic neighbor embedding.** Consider that (2) could be written as $L = \int N_y(\mathbf{z}|0, \mathbf{K}_y) \ln N_x(\mathbf{z}|0, \mathbf{K}_x) d\mathbf{z}$ where we have introduced a vector, $\mathbf{z}$, of length $N$, $\mathbf{K}_y = \frac{1}{D} \mathbf{Y} \mathbf{Y}^{\mathrm{T}}$ and we have redefined $\mathbf{K}$ as $\mathbf{K}_x$. The entropy of $N_y(\mathbf{z}|0, \mathbf{K}_y)$ is constant[7] in $\mathbf{X}$, we therefore may add it to $L$ to obtain

$$\mathrm{KL}(N_y|N_x) = \int N_y(\mathbf{z}|0, \mathbf{K}_y) \ln \frac{N_x(\mathbf{z}|0, \mathbf{K}_x)}{N_y(\mathbf{z}|0, \mathbf{K}_y)} d\mathbf{z}, \tag{5}$$

which is recognised Kullback-Leibler (KL) divergence between the two distributions. Stochastic neighbor embedding (SNE) [5] also minimises this KL divergence to visualise data. However in SNE the vector $\mathbf{z}$ is discrete.

**Generative topographic mapping.** The Generative topographic mapping [3] makes use of a radial basis function network to perform the mapping from latent space to observed space. Marginalisation of the latent space is achieved with an expectation-maximisation

(EM) algorithm. A radial basis function network is a special case of a generalised linear model and can be interpreted as a Gaussian process. Under this interpretation the GTM becomes GPLVM with a particular covariance function. The special feature of the GTM is the manner in which the latent space is represented, as a set of uniformly spaced delta functions. One could view the GPLVM as having a delta function associated with each data-point: in the GPLVM the positions of the delta functions are optimised, in the GTM each data point is associated with several different fixed delta functions.

## 4    Conclusions

We have presented a new class of models for probabilistic modelling and visualisation of high dimensional data. We provided strong theoretical grounding for the approach by proving that principal component analysis is a special case. On three real world data-sets we showed that visualisations provided by the model cluster the data in a reasonable way. Our model has an advantage over the various spectral clustering algorithms that have been presented in recent years in that, in common with the GTM, it is truly generative with an underlying probabilistic interpretation. However it does not suffer from the artificial 'discretetisation' suffered by the GTM. Our theoretical analysis also suggested a novel non-linearisation of PCA involving two kernel functions.

**Acknowledgements**   We thank Aaron Hertzmann for comments on the manuscript.

## Footnotes

[1] As can the solution for $\beta$ but since the solution for $\mathbf{W}$ is not dependent on $\beta$ we will disregard it.

[2] For independent component analysis the correct rotation matrix $\mathbf{V}$ must also be found, here we have placed no constraints on the orientation of the axes so this matrix cannot be recovered.

[3]Strictly speaking the model does not represent a mapping as a Gaussian process 'maps' to a distribution in data space rather than a point.

[4]This apparent weakness in the model may be easily rectified to allow different levels of uncertainty for each output dimension, our more constrained model allows us to visualise this uncertainty in the latent space and is therefore preferred for this work.

[5] In practice we looked for MAP solutions for all our optimisations, specifying a unit covariance Gaussian prior for the matrix $\mathbf{X}$ and using $1/\alpha$, $1/\beta$ and $1/\gamma$ for $\alpha$, $\beta$ and $\gamma$ respectively.

[6]By one-shot we mean that, given the algorithm above, each experiment was only run once with one setting of the random seed and the values of $T$ and $d$ given. If we were producing a visualisation for only one dataset this would leave us open to the criticism that our one-shot result was 'lucky'. However we present three data-sets in what follows and using a one-shot approach in problems with multiple local minima removes the temptation of preferentially selecting 'prettier' results.

[7]Computing the entropy requires $\mathbf{K}_y$ to be of full rank, this is not true in general but can be forced by adding 'jitter' to $\mathbf{K}_y$, *e.g.* $\mathbf{K}_y \rightarrow \mathbf{K}_y + \gamma^{-1} \mathbf{I}$.

## References

[1] S. Becker, S. Thrun, and K. Obermayer, editors. *Advances in Neural Information Processing Systems*, volume 15, Cambridge, MA, 2003. MIT Press.

[2] C. M. Bishop and G. D. James. Analysis of multiphase flows using dual-energy gamma densitometry and neural networks. *Nuclear Instruments and Methods in Physics Research*, A327:580–593, 1993.

[3] C. M. Bishop, M. Svensén, and C. K. I. Williams. GTM: a principled alternative to the Self-Organizing Map. In *Advances in Neural Information Processing Systems*, volume 9, pages 354–360. MIT Press, 1997.

[4] C. M. Bishop, M. Svensén, and C. K. I. Williams. GTM: the Generative Topographic Mapping. *Neural Computation*, 10(1):215–234, 1998.

[5] G. Hinton and S. Roweis. Stochastic neighbor embedding. In Becker et al. [1], pages 857–864.

[6] N. D. Lawrence, M. Seeger, and R. Herbrich. Fast sparse Gaussian process methods: The informative vector machine. In Becker et al. [1], pages 625–632.

[7] I. T. Nabney. *Netlab: Algorithms for Pattern Recognition*. Advances in Pattern Recognition. Springer, Berlin, 2001. Code available from http://www.ncrg.aston.ac.uk/netlab/.

[8] S. Roweis, L. K. Saul, and G. Hinton. Global coordination of local linear models. In T. G. Dietterich, S. Becker, and Z. Ghahramani, editors, *Advances in Neural Information Processing Systems*, volume 14, pages 889–896, Cambridge, MA, 2002. MIT Press.

[9] B. Schölkopf, A. J. Smola, and K.-R. Müller. Kernel principal component analysis. In *Proceedings 1997 International Conference on Artificial Neural Networks, ICANN'97*, page 583, Lausanne, Switzerland, 1997.

[10] M. E. Tipping. Sparse kernel principal component analysis. In T. K. Leen, T. G. Dietterich, and V. Tresp, editors, *Advances in Neural Information Processing Systems*, volume 13, pages 633–639, Cambridge, MA, 2001. MIT Press.

[11] M. E. Tipping and C. M. Bishop. Probabilistic principal component analysis. *Journal of the Royal Statistical Society, B*, 6(3):611–622, 1999.
